# Shared Segmentation of Natural Scenes Using Dependent Pitman-Yor Processes

**Erik B. Sudderth and Michael I. Jordan**
Electrical Engineering & Computer Science, University of California, Berkeley
`sudderth@cs.berkeley.edu, jordan@cs.berkeley.edu`

## Abstract

We develop a statistical framework for the simultaneous, unsupervised segmentation and discovery of visual object categories from image databases. Examining a large set of manually segmented scenes, we show that object frequencies and segment sizes both follow power law distributions, which are well modeled by the Pitman–Yor (PY) process. This nonparametric prior distribution leads to learning algorithms which discover an unknown set of objects, and segmentation methods which automatically adapt their resolution to each image. Generalizing previous applications of PY processes, we use Gaussian processes to discover spatially contiguous segments which respect image boundaries. Using a novel family of variational approximations, our approach produces segmentations which compare favorably to state-of-the-art methods, while simultaneously discovering categories shared among natural scenes.

## 1   Introduction

Images of natural environments contain a rich diversity of spatial structure at both coarse and fine scales. We would like to build systems which can automatically *discover* the visual categories (e.g., foliage, mountains, buildings, oceans) which compose such scenes. Because the "objects" of interest lack rigid forms, they are poorly suited to traditional, fixed aspect detectors. In simple cases, topic models can be used to cluster local textural elements, coarsely representing categories via a bag of visual features [1, 2]. However, spatial structure plays a crucial role in general scene interpretation [3], particularly when few labeled training examples are available.

One approach to modeling additional spatial dependence begins by precomputing one, or several, *segmentations* of each input image [4–6]. However, low-level grouping cues are often ambiguous, and fixed partitions may improperly split or merge objects. Markov random fields (MRFs) have been used to segment images into one of several known object classes [7, 8], but these approaches require manual segmentations to train category-specific appearance models. In this paper, we instead develop a statistical framework for the *unsupervised* discovery and segmentation of visual object categories. We approach this problem by considering sets of images depicting related natural scenes (see Fig. 1(a)). Using color and texture cues, our method simultaneously groups dense features into spatially coherent segments, and refines these partitions using shared appearance models. This extends the *cosegmentation* framework [9], which matches two views of a single object instance, to simultaneously segment multiple object categories across a large image database. Some recent work has pursued similar goals [6, 10], but robust object discovery remains an open challenge.

Our models are based on the *Pitman–Yor* (PY) process [11], a nonparametric Bayesian prior on infinite partitions. This generalization of the *Dirichlet process* (DP) leads to heavier-tailed, power law distributions for the frequencies of observed objects or topics. Using a large database of manual scene segmentations, Sec. 2 demonstrates that PY priors closely match the true distributions of natural segment sizes, and frequencies with which object categories are observed. Generalizing the hierarchical DP [12], Sec. 3 then describes a *hierarchical Pitman–Yor* (HPY) mixture model which shares "bag of features" appearance models among related scenes. Importantly, this approach coherently models uncertainty in the *number* of object categories and instances.

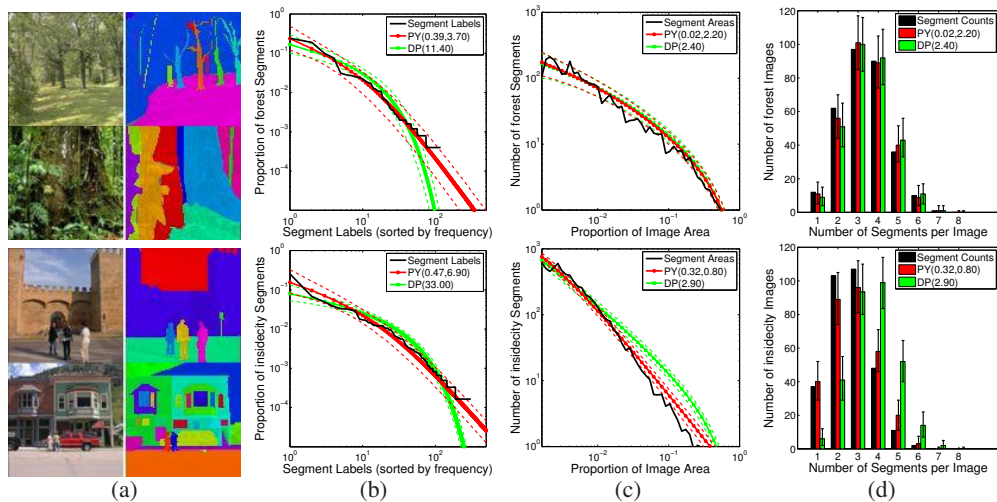

Figure 1: Validation of stick-breaking priors for the statistics of human segmentations of the *forest* (top) and *insidecity* (bottom) scene categories. We compare observed frequencies (black) to those predicted by Pitman–Yor process (PY, red circles) and Dirichlet process (DP, green squares) models. For each model, we also display 95% confidence intervals (dashed). (a) Example human segmentations, where each segment has a text label such as *sky*, *tree trunk*, *car*, or *person walking*. The full segmented database is available from LabelMe [14]. (b) Frequency with which different semantic text labels, sorted from most to least frequent on a log-log scale, are associated with segments. (c) Number of segments occupying varying proportions of the image area, on a log-log scale. (d) Counts of segments of size at least 5,000 pixels in $256 \times 256$ images of natural scenes.

As described in Sec. 4, we use *thresholded* Gaussian processes to link assignments of features to regions, and thereby produce smooth, coherent segments. Simulations show that our use of continuous latent variables captures long-range dependencies neglected by MRFs, including intervening contour cues derived from image boundaries [13]. Furthermore, our formulation naturally leads to an efficient variational learning algorithm, which automatically searches over segmentations of varying resolution. Sec. 5 concludes by demonstrating accurate segmentation of complex images, and discovery of appearance patterns shared across natural scenes.

## 2 Statistics of Natural Scene Categories

To better understand the statistical relationships underlying natural scenes, we analyze manual segmentations of Oliva and Torralba's eight categories [3]. A non-expert user partitioned each image into a variable number of polygonal segments corresponding to distinctive objects or scene elements (see Fig. 1(a)). Each segment has a semantic text label, allowing study of object co-occurrence frequencies across related scenes. There are over 29,000 segments in the collection of 2,688 images.[1]

### 2.1 Stick Breaking and Pitman–Yor Processes

The relative frequencies of different object categories, as well as the image areas they occupy, can be statistically modeled via distributions on potentially infinite *partitions*. Let $\boldsymbol{\varphi} = (\varphi_1, \varphi_2, \varphi_3, \ldots)$, $\sum_{k=1}^{\infty} \varphi_k = 1$, denote the probability mass associated with each subset. In nonparametric Bayesian statistics, prior models for partitions are often defined via a *stick-breaking* construction:

$$\varphi_k = w_k \prod_{\ell=1}^{k-1} (1 - w_\ell) = w_k \left( 1 - \sum_{\ell=1}^{k-1} \varphi_\ell \right) \qquad w_k \sim \text{Beta}(1 - \gamma_a, \gamma_b + k\gamma_a) \qquad (1)$$

This *Pitman–Yor* (PY) process [11], denoted by $\boldsymbol{\varphi} \sim \text{GEM}(\gamma_a, \gamma_b)$, is defined by two hyperparameters satisfying $0 \leq \gamma_a < 1$, $\gamma_b > -\gamma_a$. When $\gamma_a = 0$, we recover a *Dirichlet process* (DP) with concentration parameter $\gamma_b$. This construction induces a distribution on $\boldsymbol{\varphi}$ such that subsets with more mass $\varphi_k$ typically have smaller indexes $k$. When $\gamma_a > 0$, $\mathbb{E}[w_k]$ decreases with $k$, and the resulting partition frequencies follow heavier-tailed, *power law* distributions.

While the sequences of beta variables underlying PY processes lead to infinite partitions, only a random, finite subset of size $K_\varepsilon = \left| \{k \mid \varphi_k > \varepsilon\} \right|$ will have probability greater than any threshold $\varepsilon$. Implicitly, nonparametric models thus also place priors on the *number* of latent classes or objects.

---

## 2.2 Object Label Frequencies

Pitman–Yor processes have been previously used to model the well-known power law behavior of text sequences [15, 16]. Intuitively, the labels assigned to segments in the natural scene database have similar properties: some (like *sky*, *trees*, and *building*) occur frequently, while others (*rainbow*, *lichen*, *scaffolding*, *obelisk*, etc.) are more rare. Fig. 1(b) plots the observed frequencies with which unique text labels, sorted from most to least frequent, occur in two scene categories. The overlaid quantiles correspond to the best fitting DP and PY processes, with parameters $(\hat{\gamma}_a, \hat{\gamma}_b)$ estimated via maximum likelihood. When $\hat{\gamma}_a > 0$, $\log \mathbb{E}[\widetilde{\varphi}_k \mid \hat{\gamma}] \approx -\hat{\gamma}_a^{-1} \log(k) + \Delta(\hat{\gamma}_a, \hat{\gamma}_b)$ for large $k$ [11], producing power law behavior which accurately predicts observed object frequencies. In contrast, the closest fitting DP model ($\hat{\gamma}_a = 0$) significantly underestimates the number of rare labels.

We have quantitatively assessed the accuracy of these models using bootstrap significance tests [17]. The PY process provides a good fit for all categories, while there is significant evidence against the DP in most cases. By varying PY hyperparameters, we also capture interesting differences among scene types: urban, man-made environments have many more unique objects than natural ones.

## 2.3 Segment Counts and Size Distributions

We have also used the natural scene database to quantitatively validate PY priors for image partitions [17]. For natural environments, the DP and PY processes both provide accurate fits. However, some urban environments have many more small objects, producing power law area distributions (see Fig. 1(c)) better captured by PY processes. As illustrated in Fig. 1(d), PY priors also model uncertainty in the *number* of segments at various resolutions.

While power laws are often used simply as a descriptive summary of observed statistics, PY processes provide a consistent generative model which we use to develop effective segmentation algorithms. We do not claim that PY processes are the only valid prior for image areas; for example, log-normal distributions have similar properties, and may also provide a good model [18]. However, PY priors lead to efficient variational inference algorithms, avoiding the costly MCMC search required by other segmentation methods with region size priors [18, 19].

## 3 A Hierarchical Model for Bags of Image Features

We now develop *hierarchical Pitman–Yor* (HPY) process models for visual scenes. We first describe a "bag of features" model [1, 2] capturing prior knowledge about region counts and sizes, and then extend it to model spatially coherent shapes in Sec. 4. Our baseline bag of features model directly generalizes the stick-breaking representation of the hierarchical DP developed by Teh et al. [12]. N-gram language models based on HPY processes [15, 16] have somewhat different forms.

### 3.1 Hierarchical Pitman–Yor Processes

Each image is first divided into roughly 1,000 *superpixels* [18] using a variant of the normalized cuts spectral clustering algorithm [13]. We describe the texture of each superpixel via a local texton histogram [20], using band-pass filter responses quantized to $W_t = 128$ bins. Similarly, a color histogram is computed by quantizing the HSV color space into $W_c = 120$ bins. Superpixel $i$ in image $j$ is then represented by histograms $x_{ji} = (x_{ji}^t, x_{ji}^c)$ indicating its texture $x_{ji}^t$ and color $x_{ji}^c$.

Figure 2 contains a directed graphical model summarizing our HPY model for collections of local image features. Each of the potentially infinite set of global object categories occurs with frequency $\varphi_k$, where $\boldsymbol{\varphi} \sim \text{GEM}(\gamma_a, \gamma_b)$ as motivated in Sec. 2.2. Each category $k$ also has an associated appearance model $\theta_k = (\theta_k^t, \theta_k^c)$, where $\theta_k^t$ and $\theta_k^c$ parameterize multinomial distributions on the $W_t$ texture and $W_c$ color bins, respectively. These parameters are regularized by Dirichlet priors $\theta_k^t \sim \text{Dir}(\rho^t)$, $\theta_k^c \sim \text{Dir}(\rho^c)$, with hyperparameters chosen to encourage sparse distributions.

Consider a dataset containing $J$ images of related scenes, each of which is allocated an infinite set of potential segments or *regions*. As in Sec. 2.3, region $t$ occupies a random proportion $\pi_{jt}$ of the area in image $j$, where $\boldsymbol{\pi}_j \sim \text{GEM}(\alpha_a, \alpha_b)$. Each region is also associated with a particular global object category $k_{jt} \sim \boldsymbol{\varphi}$. For each superpixel $i$, we then *independently* select a region $t_{ji} \sim \boldsymbol{\pi}_j$, and sample features using parameters determined by that segment's global object category:

$$p\big(x_{ji}^t, x_{ji}^c \mid t_{ji}, \mathbf{k}_j, \boldsymbol{\theta}\big) = \text{Mult}\big(x_{ji}^t \mid \theta_{z_{ji}}^t\big) \cdot \text{Mult}\big(x_{ji}^c \mid \theta_{z_{ji}}^c\big) \qquad z_{ji} \triangleq k_{jt_{ji}} \qquad (2)$$

As in other adaptations of topic models to visual data [8], we assume that different feature channels vary independently within individual object categories and segments.

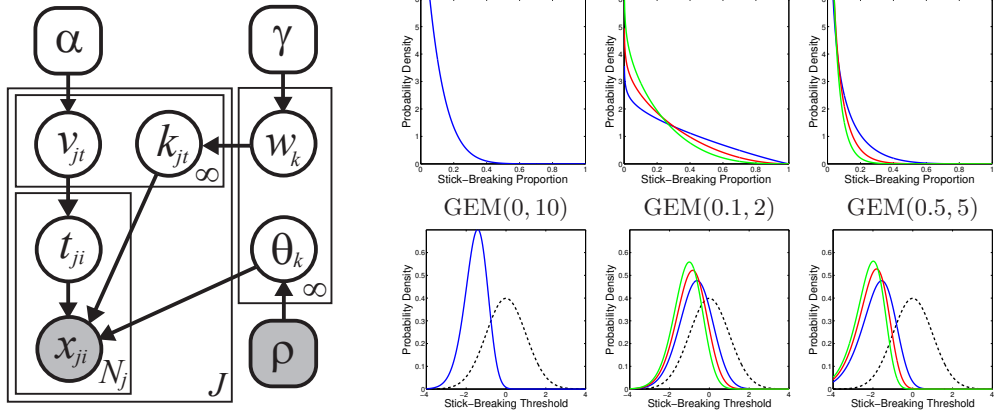

Figure 2: Stick-breaking representation of a hierarchical Pitman–Yor (HPY) model for $J$ groups of features. *Left:* Directed graphical model in which global category frequencies $\varphi \sim \text{GEM}(\gamma)$ are constructed from stick-breaking proportions $w_k \sim \text{Beta}(1 - \gamma_a, \gamma_b + k\gamma_a)$, as in Eq. (1). Similarly, $v_{jt} \sim \text{Beta}(1 - \alpha_a, \alpha_b + t\alpha_a)$ define region areas $\pi_j \sim \text{GEM}(\alpha)$ for image $j$. Each of the $N_j$ features $x_{ji}$ is independently sampled as in Eq. (2). *Upper right:* Beta distributions from which stick proportions $w_k$ are sampled for three different PY processes: $k = 1$ (blue), $k = 10$ (red), $k = 20$ (green). *Lower right:* Corresponding distributions on thresholds for an equivalent generative model employing zero mean, unit variance Gaussians (dashed black). See Sec. 4.1.

## 3.2  Variational Learning for HPY Mixture Models

To allow efficient learning of HPY model parameters from large image databases, we have developed a mean field variational method which combines and extends previous approaches for DP mixtures [21, 22] and finite topic models. Using the stick-breaking representation of Fig. 2, and a factorized variational posterior, we optimize the following lower bound on the marginal likelihood:

$$\log p(\mathbf{x} \mid \alpha, \gamma, \rho) \geq H(q) + \mathbb{E}_q[\log p(\mathbf{x}, \mathbf{k}, \mathbf{t}, \mathbf{v}, \mathbf{w}, \boldsymbol{\theta} \mid \alpha, \gamma, \rho)] \qquad (3)$$

$$q(\mathbf{k}, \mathbf{t}, \mathbf{v}, \mathbf{w}, \boldsymbol{\theta}) = \left[\prod_{k=1}^{K} q(w_k \mid \omega_k) q(\theta_k \mid \eta_k)\right] \cdot \prod_{j=1}^{J} \left[\prod_{t=1}^{T} q(v_{jt} \mid \nu_{jt}) q(k_{jt} \mid \kappa_{jt})\right] \prod_{i=1}^{N_j} q(t_{ji} \mid \tau_{ji})$$

Here, $H(q)$ is the entropy. We *truncate* the variational posterior [21] by setting $q(v_{jT} = 1) = 1$ for each image or group, and $q(w_K = 1) = 1$ for the shared global clusters. Multinomial assignments $q(k_{jt} \mid \kappa_{jt})$, $q(t_{ji} \mid \tau_{ji})$, and beta stick proportions $q(w_k \mid \omega_k)$, $q(v_{jt} \mid \nu_{jt})$, then have closed form update equations. To avoid bias, we sort the current sets of image segments, and global categories, in order of decreasing aggregate assignment probability after each iteration [22].

## 4  Segmentation with Spatially Dependent Pitman–Yor Processes

We now generalize the HPY image segmentation model of Fig. 2 to capture spatial dependencies. For simplicity, we consider a single-image model in which features $x_i$ are assigned to regions by indicator variables $z_i$, and each segment $k$ has its own appearance parameters $\theta_k$ (see Fig. 3). As in Sec. 3.1, however, this model is easily extended to share appearance parameters among images.

### 4.1  Coupling Assignments using Thresholded Gaussian Processes

Consider a generative model which partitions data into two clusters via assignments $z_i \in \{0, 1\}$ sampled such that $\mathbb{P}[z_i = 1] = v$. One representation of this sampling process first generates a Gaussian auxiliary variable $u_i \sim \mathcal{N}(0, 1)$, and then chooses $z_i$ according to the following rule:

$$z_i = \begin{cases} 1 & \text{if } u_i < \Phi^{-1}(v) \\ 0 & \text{otherwise} \end{cases} \qquad \Phi(u) \triangleq \frac{1}{\sqrt{2\pi}} \int_{-\infty}^{u} e^{-s^2/2} \, ds \qquad (4)$$

Here, $\Phi(u)$ is the standard normal *cumulative distribution function* (CDF). Since $\Phi(u_i)$ is uniformly distributed on $[0, 1]$, we immediately have $\mathbb{P}[z_i = 1] = \mathbb{P}\big[u_i < \Phi^{-1}(v)\big] = \mathbb{P}[\Phi(u_i) < v] = v$.

We adapt this idea to PY processes using the stick-breaking representation of Eq. (1). In particular, we note that if $z_i \sim \boldsymbol{\pi}$ where $\pi_k = v_k \prod_{\ell=1}^{k-1}(1 - v_\ell)$, a simple induction argument shows that $v_k = \mathbb{P}[z_i = k \mid z_i \neq k - 1, \ldots, 1]$. The stick-breaking proportion $v_k$ is thus the *conditional* probability of choosing cluster $k$, given that clusters with indexes $\ell < k$ have been rejected. Combining

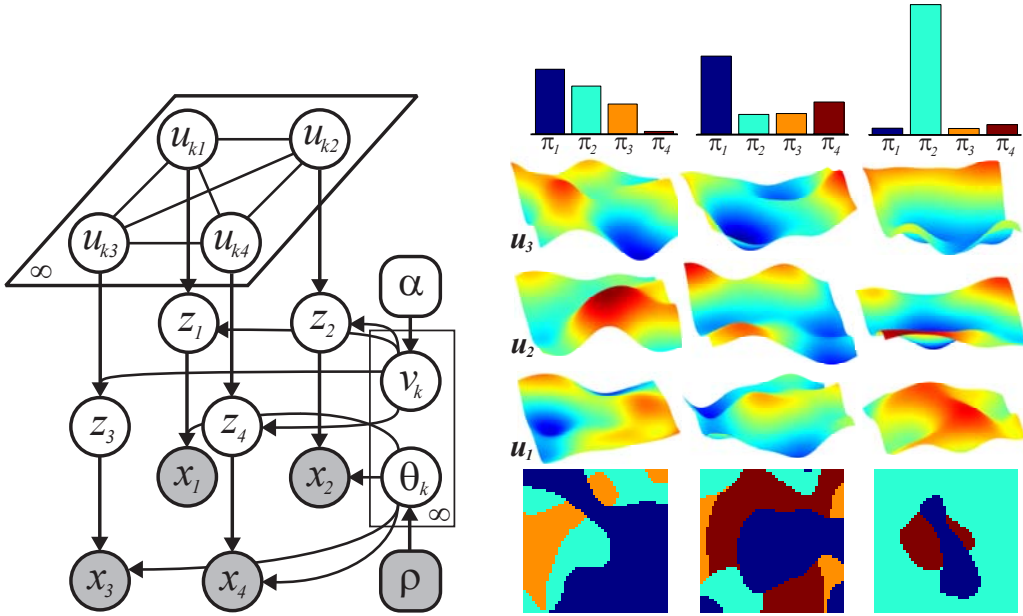

Figure 3: A nonparametric Bayesian approach to image segmentation in which thresholded Gaussian processes generate spatially dependent Pitman–Yor processes. *Left:* Directed graphical model in which expected segment areas $\pi \sim \text{GEM}(\alpha)$ are constructed from stick-breaking proportions $v_k \sim \text{Beta}(1 - \alpha_a, \alpha_b + k\alpha_a)$. Zero mean Gaussian processes ($u_{ki} \sim \mathcal{N}(0,1)$) are cut by thresholds $\Phi^{-1}(v_k)$ to produce segment assignments $z_i$, and thereby features $x_i$. *Right:* Three randomly sampled image partitions (columns), where assignments (bottom, color-coded) are determined by the *first* of the ordered Gaussian processes $\mathbf{u}_k$ to cross $\Phi^{-1}(v_k)$.

this insight with Eq. (4), we can generate samples $z_i \sim \pi$ as follows:

$$z_i = \min \left\{ k \mid u_{ki} < \Phi^{-1}(v_k) \right\} \qquad \text{where } u_{ki} \sim \mathcal{N}(0,1) \text{ and } u_{ki} \perp u_{\ell i}, k \neq \ell \qquad (5)$$

As illustrated in Fig. 3, each cluster $k$ is now associated with a zero mean *Gaussian process* (GP) $\mathbf{u}_k$, and assignments are determined by the sequence of *thresholds* in Eq. (5). If the GPs have identity covariance functions, we recover the basic HPY model of Sec. 3.1. More general covariances can be used to encode the prior probability that each feature pair occupies the same segment. Intuitively, the ordering of segments underlying this dependent PY model is analogous to *layered* appearance models [23], in which foreground layers *occlude* those that are farther from the camera.

To retain the power law prior on segment sizes justified in Sec. 2.3, we transform priors on stick proportions $v_k \sim \text{Beta}(1 - \alpha_a, \alpha_b + k\alpha_a)$ into corresponding random thresholds:

$$p(\bar{v}_k \mid \alpha) = \mathcal{N}(\bar{v}_k \mid 0, 1) \cdot \text{Beta}(\Phi(\bar{v}_k) \mid 1 - \alpha_a, \alpha_b + k\alpha_a) \qquad \bar{v}_k \triangleq \Phi^{-1}(v_k) \qquad (6)$$

Fig. 2 illustrates the threshold distributions corresponding to several different PY stick-breaking priors. As the number of features $N$ becomes large relative to the GP covariance length-scale, the proportion assigned to segment $k$ approaches $\pi_k$, where $\pi \sim \text{GEM}(\alpha_a, \alpha_b)$ as desired.

## 4.2   Variational Learning for Dependent PY Processes

Substantial innovations are required to extend the variational method of Sec. 3.2 to the Gaussian processes underlying our dependent PY processes. Complications arise due to the threshold assignment process of Eq. (5), which is "stronger" than the likelihoods typically used in probit models for GP classification, as well as the non-standard threshold prior of Eq. (6). In the simplest case, we place factorized Gaussian variational posteriors on thresholds $q(\bar{v}_k) = \mathcal{N}(\bar{v}_k \mid \nu_k, \delta_k)$ and assignment surfaces $q(u_{ki}) = \mathcal{N}(u_{ki} \mid \mu_{ki}, \lambda_{ki})$, and exploit the following key identities:

$$\mathbb{P}_q[u_{ki} < \bar{v}_k] = \Phi\left(\frac{\nu_k - \mu_{ki}}{\sqrt{\delta_k + \lambda_{ki}}}\right) \qquad \mathbb{E}_q[\log \Phi(\bar{v}_k)] \leq \log \mathbb{E}_q[\Phi(\bar{v}_k)] = \log \Phi\left(\frac{\nu_k}{\sqrt{1 + \delta_k}}\right) \qquad (7)$$

The first expression leads to closed form updates for Dirichlet appearance parameters $q(\theta_k \mid \eta_k)$, while the second evaluates the beta normalization constants in Eq. (6). We then *jointly* optimize each layer's threshold $q(\bar{v}_k)$ and assignment surface $q(\mathbf{u}_k)$, fixing all other layers, via backtracking conjugate gradient (CG) with line search. For details and further refinements, see [17].

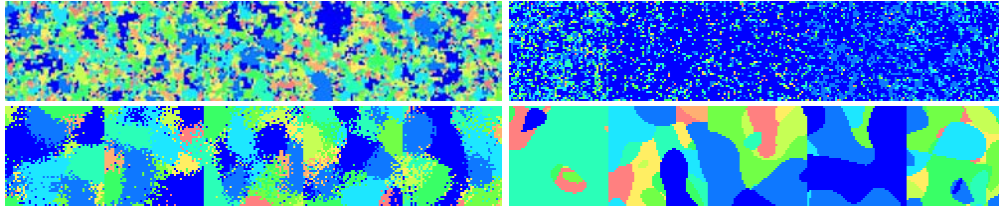

Figure 4: Five samples from each of four prior models for image partitions (color coded). *Top Left:* Nearest neighbor Potts MRF with $K = 10$ states. *Top Right:* Potts MRF with potentials biased by DP samples [28]. *Bottom Left:* Softmax model in which spatially varying assignment probabilities are coupled by logistically transformed GPs [25–27]. *Bottom Right:* PY process assignments coupled by thresholded GPs (as in Fig. 3).

### 4.3  Related Work

Recently, Duan et. al. [24] proposed a *generalized spatial Dirichlet process* which links assignments via thresholded GPs, as in Sec. 4.1. However, their focus is on modeling spatial random effects for prediction tasks, as opposed to the segmentation tasks which motivate our generalization to PY processes. Unlike our HPY extension, they do not consider approaches to sharing parameters among related groups or images. Moreover, their basic Gibbs sampler takes 12 hours on a toy dataset with 2,000 observations; our variational method jointly segments 200 scenes in comparable time.

Several authors have independently proposed a spatial model based on pointwise, multinomial logistic transformations of $K$ latent GPs [25–27]. This produces a field of smoothly varying multinomial distributions $\tilde{\pi}_i$, from which segment assignments are independently sampled as $z_i \sim \tilde{\pi}_i$. As shown in Fig. 4, this *softmax* construction produces noisy, less spatially coherent partitions. Moreover, its bias towards partitions with $K$ segments of similar size is a poor fit for natural scenes.

A previous nonparametric image segmentation method defined its prior as a normalized *product* of a DP sample $\pi \sim \mathrm{GEM}(0, \alpha)$ and a nearest neighbor MRF with Potts potentials [28]. This construction effectively treats $\log \pi$ as the *canonical*, rather than moment, parameters of the MRF, and does *not* produce partitions whose size distribution matches $\mathrm{GEM}(0, \alpha)$. Due to the phase transition which occurs with increasing potential strength, Potts models assign low probability to realistic image partitions [29]. Empirically, the DP-Potts product construction seems to have similar issues (see Fig. 4), although it can still be effective with strongly informative likelihoods [28].

## 5  Results

Figure 5 shows segmentation results for images from the scene categories considered in Sec. 2. We compare the bag of features PY model (PY-BOF), dependent PY with distance-based squared exponential covariance (PY-Dist), and dependent PY with covariance that incorporates intervening contour cues (PY-Edge) based on the $P_b$ detector [20]. The *conditionally* specified PY-Edge model scales the covariance between superpixels $i$ and $j$ by $\sqrt{1 - b_{ij}}$, where $b_{ij}$ is the largest $P_b$ response on the straight line connecting them. We convert these local covariance estimates into a globally consistent, positive definite matrix via an eigendecomposition. For the results in Figs. 5 and 6, we *independently* segment each image, without sharing appearance models or supervised training.

We compare our results to the normalized cuts spectral clustering method with varying numbers of segments (NCut($K$)), and a high-quality affinity function based on color, texture, and intervening contour cues [13]. Our PY models consistently capture variability in the number of true segments, and detect both large and small regions. In contrast, normalized cuts is implicitly biased towards regions of equal size, which produces distortions. To quantitatively evaluate results, we measure overlap with held-out human segments via the Rand index [30]. As summarized in Fig. 6, PY-BOF performs well for some images with unambiguous features, but PY-Edge is often substantially better.

We have also experimented with our hierarchical PY extension, in which color and texture distributions are shared between images. As shown in Fig. 7, many of the inferred global visual categories align reasonably with semantic categories (e.g., *sky*, *foliage*, *mountains*, or *buildings*).

## 6  Discussion

We have developed a nonparametric framework for image segmentation which uses thresholded Gaussian processes to produce spatially coupled Pitman–Yor processes. This approach produces empirically justified power law priors for region areas and object frequencies, allows visual appear-

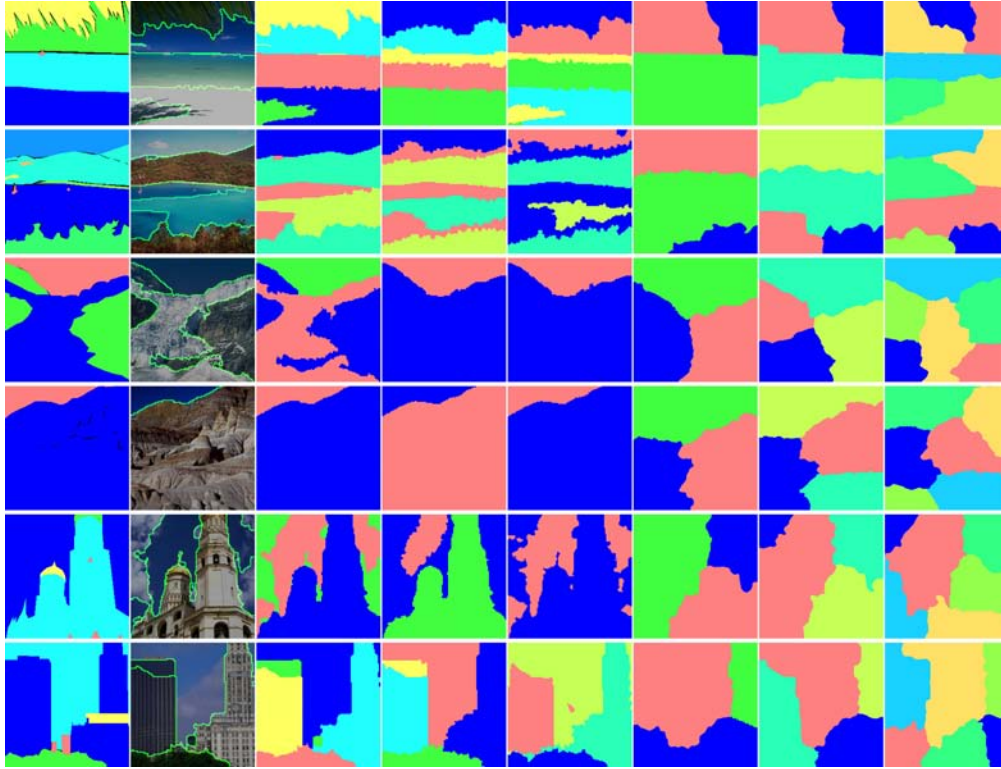

Figure 5: Segmentation results for two images (rows) from each of the *coast*, *mountain*, and *tallbuilding* scene categories. From left to right, columns show LabelMe human segments, image with boundaries inferred by PY-Edge, and segments for PY-Edge, PY-Dist, PY-BOF, NCut(3), NCut(4), and NCut(6). Best viewed in color.

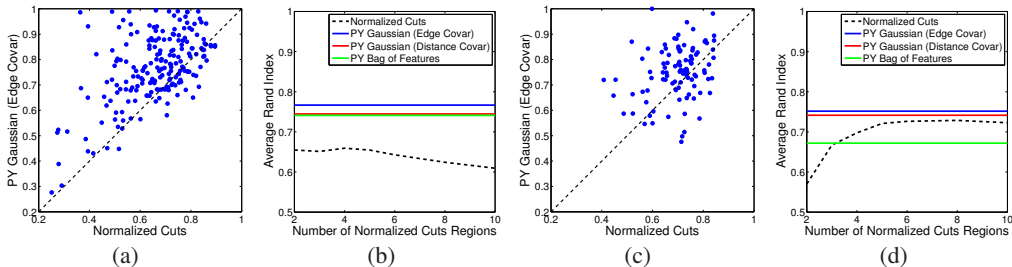

|  |  |  |  |
|:--:|:--:|:--:|:--:|
| (a) | (b) | (c) | (d) |

Figure 6: Quantitative comparison of segmentation results to human segments, using the Rand index. (a) Scatter plot of PY-Edge and NCut(4) Rand indexes for 200 *mountain* images. (b) Average Rand indexes for *mountain* images. We plot the performance of NCut($K$) versus the number of segments $K$, compared to the variable resolution segmentations of PY-Edge, PY-Dist, and PY-BOF. (c) Scatter plot of PY-Edge and NCut(6) Rand indexes for 200 *tallbuilding* images. (d) Average Rand indexes for *tallbuilding* images.

ance models to be flexibly shared among natural scenes, and leads to efficient variational inference algorithms which automatically search over segmentations of varying resolution. We believe this provides a promising starting point for discovery of shape-based visual appearance models, as well as weakly supervised nonparametric learning in other, non-visual application domains.

**Acknowledgments**    We thank Charless Fowlkes and David Martin for the $P_b$ boundary estimation and segmentation code, Antonio Torralba for helpful conversations, and Sra. Barriuso for her image labeling expertise. This research supported by ONR Grant N00014-06-1-0734, and DARPA IPTO Contract FA8750-05-2-0249.

## References

[1] L. Fei-Fei and P. Perona. A Bayesian hierarchical model for learning natural scene categories. In *CVPR*, volume 2, pages 524–531, 2005.
[2] J. Sivic, B. C. Russell, A. A. Efros, A. Zisserman, and W. T. Freeman. Discovering objects and their location in images. In *ICCV*, volume 1, pages 370–377, 2005.
[3] A. Oliva and A. Torralba. Modeling the shape of the scene: A holistic representation of the spatial envelope. *IJCV*, 42(3):145–175, 2001.

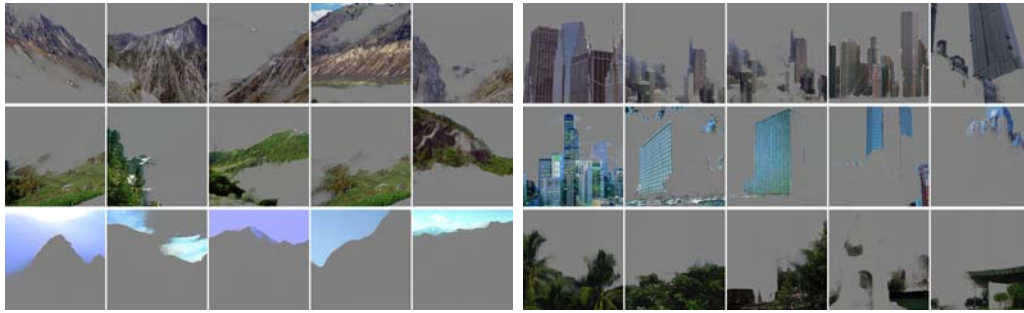

Figure 7: Most significant segments associated with each of three shared, global visual categories (rows) for hierarchical PY-Edge models trained with 200 images of *mountain* (left) or *tallbuilding* (right) scenes.

[4] L. Cao and L. Fei-Fei. Spatially coherent latent topic model for concurrent object segmentation and classification. In *ICCV*, 2007.

[5] B. C. Russell, A. A. Efros, J. Sivic, W. T. Freeman, and A. Zisserman. Using multiple segmentations to discover objects and their extent in image collections. In *CVPR*, volume 2, pages 1605–1614, 2006.

[6] S. Todorovic and N. Ahuja. Learning the taxonomy and models of categories present in arbitrary images. In *ICCV*, 2007.

[7] X. He, R. S. Zemel, and M. A. Carreira-Perpiñán. Multiscale conditional random fields for image labeling. In *CVPR*, volume 2, pages 695–702, 2004.

[8] J. Verbeek and B. Triggs. Region classification with Markov field aspect models. In *CVPR*, 2007.

[9] C. Rother, V. Kolmogorov, T. Minka, and A. Blake. Cosegmentation of image pairs by histogram matching: Incorporating a global constraint into MRFs. In *CVPR*, volume 1, pages 993–1000, 2006.

[10] M. Andreetto, L. Zelnik-Manor, and P. Perona. Non-parametric probabilistic image segmentation. In *ICCV*, 2007.

[11] J. Pitman and M. Yor. The two-parameter Poisson–Dirichlet distribution derived from a stable subordinator. *Ann. Prob.*, 25(2):855–900, 1997.

[12] Y. W. Teh, M. I. Jordan, M. J. Beal, and D. M. Blei. Hierarchical Dirichlet processes. *J. Amer. Stat. Assoc.*, 101(476):1566–1581, December 2006.

[13] C. Fowlkes, D. Martin, and J. Malik. Learning affinity functions for image segmentation: Combining patch-based and gradient-based approaches. In *CVPR*, volume 2, pages 54–61, 2003.

[14] B. C. Russell, A. Torralba, K. P. Murphy, and W. T. Freeman. LabelMe: A database and web-based tool for image annotation. *IJCV*, 77:157–173, 2008.

[15] S. Goldwater, T. L. Griffiths, and M. Johnson. Interpolating between types and tokens by estimating power-law generators. In *NIPS 18*, pages 459–466. MIT Press, 2006.

[16] Y. W. Teh. A hierarchical Bayesian language model based on Pitman–Yor processes. In *Coling/ACL*, 2006.

[17] E. B. Sudderth and M. I. Jordan. Shared segmentation of natural scenes using dependent Pitman-Yor processes. Technical report, Dept. of Statistics, University of California, Berkeley. In preparation, 2009.

[18] X. Ren and J. Malik. Learning a classification model for segmentation. In *ICCV*, 2003.

[19] Z. Tu and S. C. Zhu. Image segmentation by data-driven Markov chain Monte Carlo. *IEEE Trans. PAMI*, 24(5):657–673, May 2002.

[20] D. R. Martin, C. C. Fowlkes, and J. Malik. Learning to detect natural image boundaries using local brightness, color, and texture cues. *IEEE Trans. PAMI*, 26(5):530–549, May 2004.

[21] D. M. Blei and M. I. Jordan. Variational inference for Dirichlet process mixtures. *Bayes. Anal.*, 1(1):121–144, 2006.

[22] K. Kurihara, M. Welling, and Y. W. Teh. Collapsed variational Dirichlet process mixture models. In *IJCAI 20*, pages 2796–2801, 2007.

[23] J. Y. A. Wang and E. H. Adelson. Representing moving images with layers. *IEEE Trans. IP*, 3(5):625–638, September 1994.

[24] J. A. Duan, M. Guindani, and A. E. Gelfand. Generalized spatial Dirichlet process models. *Biometrika*, 94(4):809–825, 2007.

[25] C. Fernández and P. J. Green. Modelling spatially correlated data via mixtures: A Bayesian approach. *J. R. Stat. Soc. B*, 64(4):805–826, 2002.

[26] M. A. T. Figueiredo. Bayesian image segmentation using Gaussian field priors. In *CVPR Workshop on Energy Minimization Methods in Computer Vision and Pattern Recognition*, 2005.

[27] M. W. Woolrich and T. E. Behrens. Variational Bayes inference of spatial mixture models for segmentation. *IEEE Trans. MI*, 25(10):1380–1391, October 2006.

[28] P. Orbanz and J. M. Buhmann. Smooth image segmentation by nonparametric Bayesian inference. In *ECCV*, volume 1, pages 444–457, 2006.

[29] R. D. Morris, X. Descombes, and J. Zerubia. The Ising/Potts model is not well suited to segmentation tasks. In *IEEE DSP Workshop*, 1996.

[30] R. Unnikrishnan, C. Pantofaru, and M. Hebert. Toward objective evaluation of image segmentation algorithms. *IEEE Trans. PAMI*, 29(6):929–944, June 2007.
